# Assessing Blinding in Clinical Trials

Ognjen Arandjelović
Deakin University, Australia

## Abstract

The interaction between the patient's expected outcome of an intervention and the inherent effects of that intervention can have extraordinary effects. Thus in clinical trials an effort is made to conceal the nature of the administered intervention from the participants in the trial i.e. to *blind* it. Yet, in practice perfect blinding is impossible to ensure or even verify. The current standard is follow up the trial with an auxiliary questionnaire, which allows trial participants to express their belief concerning the assigned intervention and which is used to compute a measure of the extent of blinding in the trial. If the estimated extent of blinding exceeds a threshold the trial is deemed sufficiently blinded; otherwise, the trial is deemed to have failed. In this paper we make several important contributions. Firstly, we identify a series of fundamental problems of the aforesaid practice and discuss them in context of the most commonly used blinding measures. Secondly, motivated by the highlighted problems, we formulate a novel method for handling imperfectly blinded trials. We too adopt a post-trial feedback questionnaire but interpret the collected data using an original approach, fundamentally different from those previously proposed. Unlike previous approaches, ours is void of any *ad hoc* free parameters, is robust to small changes in auxiliary data and is not predicated on any strong assumptions used to interpret participants' feedback.

## 1 Introduction

Ultimately, the main aim of a clinical trial is straightforward: it is to examine and quantify the effectiveness of a treatment of interest. Effectiveness is evaluated relative to the effectiveness of a particular reference, the so-called *control* intervention. To ensure that the aforementioned comparison is meaningful, it is of essential importance to ensure that any factors not inherently associated with the two interventions (treatment and control) are normalized (controlled) between the two groups. This ensures that the observed differential outcome truly is the effect of differing interventions rather than any orthogonal, confounding variables. A related challenge is that of *blinding*. Blinding refers to the concealment of the type of administered intervention from the individuals/patients participating in a trial and its aim is to eliminate differential placebo effect between groups [10, 3, 11]. Although conceptually simple, the problem of blinding poses difficult challenges in a practical clinical setup. We highlight two specific challenges which most strongly motivate the work of the present paper. The first of these stems from the difficulty of ensuring that absolute blinding with respect to a particular trial variable is achieved. The second challenge arises as a consequence of the fact that blinding can only be attempted with respect to those variables of the trial which have been identified as revealing of the treatment administered. Put differently, it is always possible that a particular variable which can reveal the nature of the treatment to a trial participant is not identified by the trial designers and thus that no blinding with respect to it is attempted or achieved. This is a ubiquitous problem, present in every controlled trial, and one which can severely affect the trial's outcome.

Given that it is both practically and in principle impossible to ensure perfect blinding, the practice of *post hoc* assessment of the level of blinding achieved has been gaining popularity and general acceptance by the clinical community. The key idea is to use a statistical model and the participants' responses to a generic post-trial questionnaire to quantify the participants' knowledge about the administered intervention. While the statistical model used to this end has been a source of disagreement between researchers, as discussed in detail in Sec 2, the general approach is shared by different methods described in the literature. In this paper we argue that this common approach suffers from several important limitations. Motivated by these, in the present work we propose a novel statistical framework and use it to derive an original method for integrated trial assessment which is experimentally shown to produce more meaningful and more clearly interpretable data.

Table 1: Notational convention for mathematical symbols adopted in this paper.

| Symbol | Description |
|---|---|
| $a$ | subscript specifying group assignment; $a = C$ and $a = T$ signify control and treatment groups |
| $g$ | subscript specifying membership belief; $g = -$ and $g = +$ signify belief in control and treatment group memberships, $g = 0$ signifies uncertainty |
| $P_{ag}$ | proportion of participants who were assigned to group $a$ and believe the membership to be $g$ |
| $P_a$ | proportion of participants who were assigned assigned to group $a$ |
| $P_g$ | proportion of participants who believe their group membership to be $g$ |

## 2 Previous Work

In this section we describe the general methodology of auxiliary post-trial data collection, the two most influential statistical models which use the aforesaid data to quantify the extent of blinding in a trial, and discuss the key limitations of the existing approaches which motivate the work described in the present paper.

### 2.1 Method 1: James's Blinding Index

At the heart of the so-called *blinding index* proposed by James *et al.* [7] is the observation that the effect of a particular intervention is affected by the participant's perception of the effectiveness of the intervention the participant believes was administered. For example, a control group member who incorrectly believes to be a member of the treatment group may indeed experience positive effects expected from the studied treatment. The is the extensively studied placebo effect [2, 9].

**Auxiliary Data**  James *et al.* propose the use of a post-trial questionnaire to assess the level of blinding in a trial. The participants are asked if they believe that they were assigned to the (i) control or (ii) treatment groups, or (iii) if they are uncertain of their assignment (the "don't know" response). Extensions of this scheme which attempt to harness more detailed information have also been used, e.g. allowing the participants to quantify the strength of their belief.

**Blinding Level**  The existing work on the assessment of trial blinding uses the collected auxiliary data to calculate a statistic referred to as the blinding index. For a 3-tier auxiliary questionnaire, James *et al.* [7] define their index as (our mathematical notation is summarized in Tab 1):

$$\rho_1 = \frac{1}{2} \Big[ 1 + P_0 + (1 - P_0) \cdot \Delta \Big] \tag{1}$$

It can attain values in the interval $[0, 1]$, higher values denoting increasing level of blindness. Thus $\rho_1 = 1$ indicates perfect blinding and $\rho_1 = 0$ an unblinded trial. The statistic $\Delta$ takes into account the distribution of participants who have a decisive belief regarding their assignment:

$$\Delta = \sum_{a \in \{P,T\}} \sum_{g \in \{+,-\}} \omega_{ag} \frac{P_{ag}(1 - P_0) - P_g(P_a - P_{a0})}{(1 - P_0)^2} \tag{2}$$

The constants $\omega_{ag}$ are weighting coefficients whose effect is to scale relative contributions of the correct and incorrect assignment guesses. To gain intuitive insight into the nature of $\rho_1$, consider the plot shown in Fig 1(a). It is readily apparent that $\rho_1$ is a concave function which attains its maximal value of 1 when (i) all participants are uncertain of their assignment or (ii) when all participants have an incorrect belief regarding their assignment.

In comparison with the case of $P_0 = 1$ the attainment of the maximal value $\rho_1 = 1$ for $P_{T+} = P_{C-} = 0$ is more questionable. While it is tempting to reason that blinding must have been successful since no participant correctly guessed their assignment, it would be erroneous to do so. In particular, the *consistency* of the wrong belief amongst trial participants actually reveals unblinding, but with the participants' incorrect association of the unblinded factor with the corresponding group assignment. For example, the treatment may cause perceivable side effects (thus unblinding the participants) and the worsening of the condition of the treatment group participants. This observation which could lead them to the conclusion that they were assigned to the control group.

### 2.2 Method 2: Bang's Blinding Index

The blinding index $\rho_1$ places a lot of value on those participants who plead ignorance regarding their assignment status. Bang *et al.* argue that the non-decisive "don't know" response may not express a

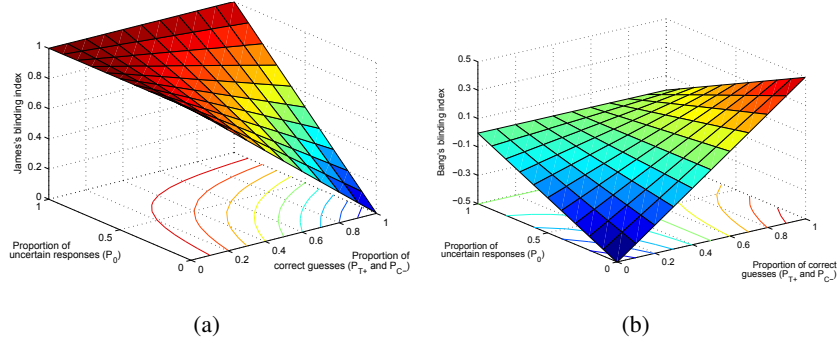

(a)             (b)

Figure 1: Dependency of the blinding indexes (a) $\rho_1$ and (b) $\rho_2'$ on the proportions of "don't know" responses $P_0$, and the correct assignment guesses $P_{T+}$ and $P_{C-}$. Note that although $P_{T+}$ and $P_{C-}$ are independent variables, due to their symmetric contributions and for the purpose of easier visualization, in this plot it was taken that $P_{T+}$ and $P_{C-}$ were always equal.

true lack of knowledge but rather that it may be a conservative response born out of desire to appear balanced in judgement [1]. Thus, they propose an alternative which instead most heavily weights the contribution of *decisive responses*. Because decisive responses can be in either the positive or the negative direction, the index is asymmetrical and can be applied separately to treatment and control groups. For a 3-tier auxiliary questionnaire, the index for the treatment group is defined as:

$$\rho_2' = \left( 2 \, \frac{P_{C-}}{P_{C-} + P_{T-}} - 1 \right) \cdot \frac{P_{T-} + P_{T+}}{\sum_{g \in -,0,+} P_{Tg}}. \tag{3}$$

The behaviour of $\rho_2'$ can be seen in Fig 1(b) which plots it against the proportions of indecisive responses and correct guesses. It is readily apparent that the plot has a form very different from that in Fig 1(a) showing the corresponding variation of $\rho_1$. Firstly, note that unlike $\rho_1$, the range of values for $\rho_2$ is $[-0.5, 0.5]$. The value of $\rho_2 = 0$ indicates perfect blinding, $\rho_2 = 0.5$ an unblinded trial and $\rho_2 = -0.5$ an unblinded trial with incorrect assignment association, as discussed in Sec 2.1.

As the plot shows, this index achieves its perfect blinding value only when $P_0 = 1$. Unlike $\rho_1$, the case when $P_{T+} = P_{C-} = 0$ does not necessarily result in perfect blinding. Also, $P_{T+} = P_{C-} = 1$ and $P_0 = 0$ deems the trial unblinded, as does $P_{T+} = P_{C-} = 0$ and $P_0 = 0$ but with the incorrect assignment association. Contrast this with the corresponding value of $\rho_1$.

## 3    Limitations of the Current Best Standards

In the preceding sections we described two blinding indexes most widely used in practice to assess the level of blinding in controlled clinical trials. To highlight and motivate the contribution of the present work, we now analyze the limitations of the aforesaid approaches.

**Adjustment of Free Parameters**    One of the most obvious difficulties encountered when applying either of the described blinding indexes concerns the need to choose appropriate values for the free parameters in Equations (2) and (3) in their general form. These are the weighting constants $w_{ag}$. Recall that their purpose is to scale the relative contributions of different responses. Although not without an intuitive appeal, a thorough analysis of this *ad hoc* approach reveals a series of problems, both inherent and practical. Firstly, there is no objective underlying mechanism which would explain why the contributions of different responses should be combined linearly at all. What is more, even if linear combination is adopted, it is fundamentally the case that there is no principled method of choosing the values of the weighting constants – the lack of observable "ground truth" means that it is not possible to objectively compare the quality of different predictions. Lastly, the values of "best" weighting constant ratios are likely to differ from trial to trial.

**Interpretation of Participants' Feedback**    It is important to highlight that both the index of James *et al.* as well as that of Bang *et al.* use the same type of feedback data collected from the trial participants – the participants' stated belief regarding their trial group assignment and their degree of confidence. Where the two approaches differ in is the *interpretation* of the participants' answers.

James *et al.* interpret the non-decisive, "don't know" response as indicative of true lack of knowledge regarding the nature of the intervention (treatment or control). If the trial participants are ignorant of their group assignment, it is assumed that they have indeed been blinded. Consequently, $\rho_1$ heavily relies on the proportion of the non-decisive participants. However, the "don't know" response may

not truly represent lack of knowledge. Instead, this response may be seen as a conservative one, reflecting the participants' desire to appear balanced in their judgement or indeed the response that the participants believe would please the trial administration staff. Thus, $\rho'_2$ mostly relies on the responses of those trial participants who did express belief regarding their group assignment. Blindness is measured by comparing the observed statistics of decisive responses with those expected from an ideal, fully blinded trial. However, this interpretation of participants' responses is readily criticized too. As Hemiliä amongst others notes, because the participants' feedback is collected *post hoc* it is possible that even a perfectly blinded subject becomes aware of the correct assignment by virtue of observing the effects (or lack thereof) of the assigned intervention [5]. Considering the same issue, Henneicke-von Zepelin [6] suggested that auxiliary data should be collected before or shortly after the commencement of a trial. However, this is in most cases unsatisfactory as the participants would not have yet been exposed to any unblinded aspects of the trial. As we demonstrate in the next section, the approach proposed in this paper entirely avoids this problem.

**Sensitivity to Small Input Differences**   Both James *et al.* and Bang *et al.* establish the level of blindness in a trial by computing a blinding index and then comparing it with a predefined threshold. This hard thresholding whereby a trial is considered either sufficiently well blinded or not means that the outcome of the blinding assessment can exhibit high sensitivity to small differences in participants' responses. The response of a single individual may change the assessment outcome. Yet, such binarization in some form is necessitated by the nature of the blinding indexes because neither of the two described statistics has a clear practical interpretation in the clinical context. The task of choosing the value of the aforesaid threshold suffers from much the same problems as the task of selecting the values of the weighting constants, discussed previously – inherently, there is no objective and meaningful way of defining the optimal threshold value, and the value actually selected by the practitioner is likely to vary from trial to trial.

**Inference Atomization**   The problem of high sensitivity to small input differences considered previously is but one of the consequences of the *inference atomization*. Specifically, observe that the analysis of the trial outcome data is separated from the blinding assessment. Indeed, only if the trial is deemed sufficiently well blinded does the analysis of actual trial data proceed. Thus, if the blinding index falls short of the predetermined threshold, the data is effectively thrown away and the trial needs to be repeated. On the other hand, if the blinding index exceeds the threshold, the analysis of data is performed in the same manner regardless of the actual value of the index, that is, regardless of whether it is just above the threshold or if it indicates perfect blinding.

The variety of problems that emerges from the atomization of different statistical aspects of a trial is inherently rooted in the very nature of the framework adopted by James *et al.* and Bang James *et al.* alike. As stated earlier, neither of the two indexes has a clear practical interpretation in the clinical context. For example, neither tells the clinician the probability that a particular portion of the participants were unblinded, nor the probability of a particular level of unblinding. Instead, from the point of view of a clinician, the blinding index behaves like a black box which deems the trial well blinded or not, with little additional insight.

## 4   Principled Approach to Controlled Clinical Trial Data Analysis

We now describe a principled method for inference from collected trial data.

### 4.1   Study Design and Outcome Model

As we demonstrated in the previous section, many of the problems of the approaches proposed by James *et al.* and Bang *et al.* inherently stem from the underlying statistical model. Although our approach uses the same type of participants' feedback data, our statistical model differs significantly from that employed in previous works.

In the general case, the effectiveness of a particular intervention in a trial participant depends on the inherent effects of the intervention, as well as the participant's expectations (conscious or not). Thus, in the interpretation of trial results, we separately consider each population of participants which share the same combination of the type of intervention and the expressed belief regarding this group assignment. This is conceptually illustrated in Fig 2.

A key idea of the proposed method is that because the outcome of an intervention depends on both the inherent effects of the intervention and the participants' expectations, the effectiveness should be inferred in a like-for-like fashion. In other words, the response observed in, say, the sub-group of participants assigned to the control group whose feedback professes belief in the control group

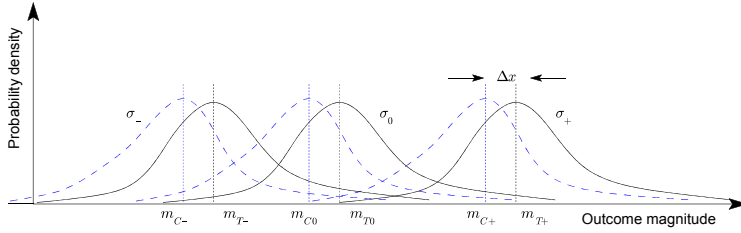

Figure 2: Conceptual illustration of the proposed statistical model for the 3-tier feedback questionnaire. Dotted and solid lines show respectively the probability density functions of the measured trial outcome across individuals in the three control and treatment sub-groups.

assignment should be compared with the response of only the sub-group of the treatment group who equally professed belief in the control group assignment. Similarly, the "don't know" sub-groups should be compared only with each other, as should the subgroups corresponding to the belief in the treatment assignment. This idea is formalized next.

## 4.2 Inference

Consider two corresponding sub-groups, that is, sub-groups corresponding to different types of received intervention but the same response in the participants' feedback questionnaire. Furthermore, let the benefit of an intervention observed in a particular participant be expressed as a real number $x_{ag}^{(i)}$. Thus, and without loss of generality, a greater $x_{ag}^{(i)}$ indicates greater benefit. For example, $x_i$ may represent the amount of fat loss in a fat loss trial, the reduction in blood plasma LDL in a statin trial etc. Our goal is to infer $p(\Delta x)$, that is, the probability density function over the difference $\Delta x$ in the benefit observed across the two compared sub-groups.

Let $D_{Cg} = \{x_{Cg}^{(1)}, \ldots, x_{Cg}^{(n_{Cg})}\}$ be the trial outcome data collected from a control sub-group and $D_{Tg} = \{x_{Tg}^{(1)}, \ldots, x_{Tg}^{(n_{Tg})}\}$ of the matching treatment sub-group. Then, if $D_g = D_{Cg} \cup D_{Tg}$ is the totality of all data of participants who believe they were assigned to the group $g$:

$$p(\Delta x \mid D_g) = \frac{P(D_g \mid \Delta x)\, p(\Delta x)}{p(D_g)}. \tag{4}$$

Modelling the response of each sub-group using a normal distribution

$$x_{Cg}^{(i)} \sim \mathcal{N}(m_{Cg}, \sigma_g) \qquad \text{and} \qquad x_{Tg}^{(j)} \sim \mathcal{N}(m_{Tg}, \sigma_g) \tag{5}$$

and remembering that for the underlying distributions it holds that $m_{Cg} + \Delta x = m_{Tg}$, allows us to further write

$$p(\Delta x \mid D_k) \propto p(D_g \mid \Delta x) = \int_{m_{Cg}} \int_{\sigma_g} p(D_g \mid \Delta x, m_{Cg}, \sigma_g)\, p(m_{Cg})\, p(\sigma_g)\, d\sigma_g\, dm_{Cg} \tag{6}$$

where $p(m_{Cg})$ is a prior on the mean of the control sub-group and $p(\sigma_g)$ a prior on the standard deviation within sub-groups. What Eq (6) expresses is the process of probability density function marginalization over nuisance variables $m_{Cg}$ and $\sigma_g$. Since the values of these latent model variables are unknown, marginalization takes into account all of the possibilities and weights them in proportion to the supporting evidence.

When two corresponding sub-groups of participants are considered, for uninformed priors over $m_{Cg}$ and $\sigma_g$, the posterior distribution of $\Delta x$ is given by:

$$p(\Delta x \mid D_g) \propto c_g^{-\frac{n_{Cg}+n_{Tg}-1}{2}} = c_g^{-\frac{n_g-1}{2}} \tag{7}$$

where constant scaling factors have been omitted for clarity, and

$$c_g = \sum_{i=1}^{n_{Cg}} x_{Cg}^{(i)}{}^2 + \sum_{j=1}^{n_{Tg}} (x_{Tg}^{(j)} + \Delta x)^2 - \left[\sum_{i=1}^{n_{Cg}} x_{Cg}^{(i)} + \sum_{j=1}^{n_{Tg}} (x_{Tg}^{(j)} + \Delta x)\right]^2 / (n_{Cg} + n_{Tg}) \tag{8}$$

Extending to the joint inference over the entire data corpus, the posterior can be computed simply as a product of all sub-group pair posteriors (up to a scaling constant):

$$p(\Delta x \mid \cup_g D_g) \propto \prod_g p(\Delta x \mid D_g) \propto \prod_g c_g^{-\frac{n_g-1}{2}} \tag{9}$$

The estimate of the posterior distribution of $\Delta x$ in Eq (9) is the best estimate that can be made using the available data, and it is of the most interest to the clinician. However, as we will discuss in Sec 6, both Eq (7) and (9) have significance in the interpretation of trial results and their joint consideration can be used to reveal important additional information about the effectiveness of the treatment.

## 5   Experiments

Certain advantages of the proposed methodology over previous approaches are *ipso facto* inherent in the theory, e.g. the absence of free parameters. Other claimed properties of the method, such as its robustness to small input differences, are not immediately obvious. In this section we present the results of a series of experiments which demonstrate the superiority of the proposed method.

### 5.1   Evaluation Methodology

In contrast to the methods of James *et al.* and Bang *et al.* which do not attempt to infer any objective and measurable quantity, the proposed approach pools all available data (trial outcomes and auxiliary questionnaire feedback) in an effort to evaluate robustly the effectiveness of the studied treatment. This feature of our method allows us to directly evaluate its performance. Specifically, we employ a computer-based simulation whereby data is first randomly (or rather pseudo-randomly) generated using a statistical model with adjustable parameters, followed by the application of the proposed method which is used to infer the said parameters. The values inferred by our method can then be directly compared with their known true values.

**Exp 1: Reference**   For our first experiment, we simulated a trial involving 200 individuals, half of which were assigned to the control and half to the treatment group. For each of the groups, 60% of the participants were taken to be in the "undecided" subgroups $G_{C0}$ and $G_{T0}$. The remaining 40% of the participants was split between correct and incorrect guesses of the assigned intervention in proportion $3:1$. In this initial experiment we assume that all participants correctly disclosed their belief regarding which group they were assigned to. Note that this assumption is done purely in the process of generating data for the experiment – neither this nor any of the preceding information is used by our method to analyze the outcome of the trial.

We set the differential effect of treatment to $\Delta x = 0.1$ and the standard deviation of variability within each of the assignment-response subgroups to $\sigma_- = \sigma_0 = \sigma_+ = 0.1$. Relative to genuine lack of belief in either control or treatment group assignments, belief in control group assignment was set to exhibit negative effect of magnitude $0.2$ and that in treatment group assignment a positive effect of magnitude $0.2$. Intervention outcomes were then generated by repeated random draws from the corresponding distributions. For example, the outcome associated with a participant in $G_{C-}$ was determined by a random draw from the normal distribution $\mathcal{N}(m_{C-}, \sigma_-)$.

The result of applying the proposed method is summarized in Fig 3 which plots the posteriors (bold lines) corresponding to the three subgroups matched by the patients' post-trial belief and the amalgamated posterior. The *maximum a posteriori* (MAP) value of the estimate of the differential effectiveness of the treatment is $\Delta x^* \approx 0.107$, which is close to the true value of $\Delta x = 0.1$. In comparison, when the differential effectiveness is estimated by subtracting the mean response of the control group from that of the treatment group, without the use of our matching sub-groups based statistical model, the estimate is $\overline{\Delta x} \approx 0.141$. Finally, the corresponding values of the blinding indices proposed by James *et al.* and Bang *et al.* are $\rho_1 = 0.53$ and $\rho_2' = \rho_2'' = 0.10$. Notice that the former indicates a level of blinding roughly half way between a perfectly blinded and unblinded trial, while the latter deems the trial nearly perfectly blinded.

### 5.1.1   Exp 2: Conservative Distortion

We modify the baseline experiment by simulating conservative behavioural tendency of participants in a trial. This was achieved by randomly choosing individuals from decisive subgroups and re-assigning them to their corresponding indecisive subgroup *without* changing their treatment's observed effectiveness. The probability of re-assignment was set to $p_{cons} = 0.2$.

As before, we applied the proposed method on the modified data and display the key results in Fig 3. In addition to the new subgroup posteriors (dotted lines), for comparison in Fig 3(a) we also show the three initial subgroup posteriors from Exp 1 (solid lines). The baseline (thick solid line) and new (thin solid line) amalgamated posteriors are shown in Fig 3(b). Fig 3(b) also shows the semi-amalgamated posterior obtained using only decisive subgroups which, by experimental design,

comprise data of only those individuals which honestly disclosed their belief of group assignment. The new MAP value for the differential effectiveness using the amalgamated posterior can be seen to be $\Delta x^* \approx 0.122$ and that using the semi-amalgamated posterior $\Delta x^* \approx 0.116$. In Sec 6 we will show how the difference in statistical features of sub-group posteriors can be used to select the most reliable posteriors to amalgamate, as well as to reveal additional insight into the nature of the studied treatment and the blinding in the trial.

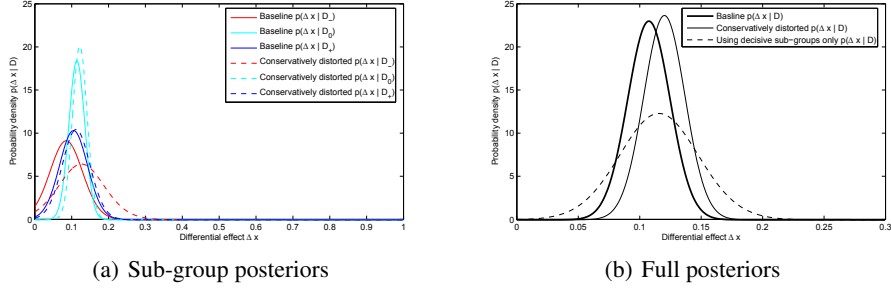

| (a) Sub-group posteriors | (b) Full posteriors |
|---|---|

Figure 3: Exp 2: (a) Posteriors for the differential effect treatment computed using the data $D_g$ of each experimental sub-group comprising control and treatment individuals matched by their feedback. (b) Posterior for the differential effect treatment computed using all available data.

**Exp 3: Asymmetric Progressive Unblinding**   Starting with the baseline setup, we simulate unblinding of previously undecided individuals of the treatment group. In other words, in each turn we re-assign an individual from the subgroup $G_{T0}$ to the subgroup $G_{T+}$ and compute the novel distribution for $\Delta x$.

The robustness of our method is illustrated in Fig 4(a), which shows the MAP estimate of the effectiveness of the treatment after an increasing number of participants were unblinded. This estimate only shows small random perturbations, with the corresponding standard deviation of $0.0054$. The plots in Fig 4(b) show the variation of the two blinding indexes throughout the experiment. As expected from the change in the participants' auxiliary data, both indexes change in value dramatically. The index of James *et al.* decreases, while that of Bang *et al.* increases in absolute value, indicating agreement on the lowered level of blinding.

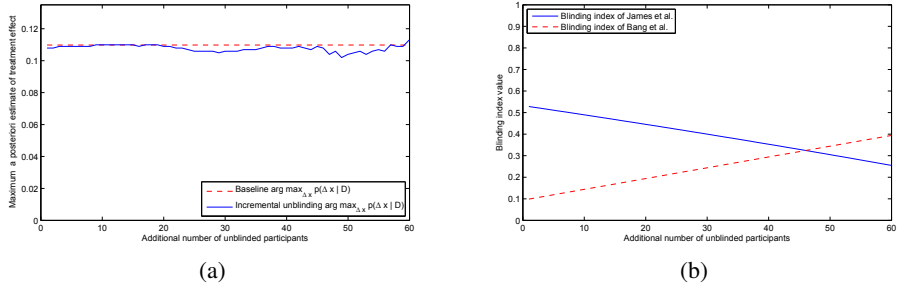

| (a) | (b) |
|---|---|

Figure 4: Exp 3: (a) The MAP estimate of the treatment effectiveness as the participants assigned to the treatment group are progressively unblinded. (b) The values of the blinding indexes $\rho_1$ (blue line) and $\rho_2'$ (red line), computed at each step of the progressive unblinding of the participants assigned to the treatment group.

**Exp 4: Symmetric Progressive Unblinding**   As in Exp 3 we start with the baseline setup and simulate unblinding of previously undecided individuals of the treatment group. In each turn we re-assign an individual from $G_{T0}$ to $G_{T+}$ and an individual from $G_{C0}$ to $G_{C-}$, and compute the novel distribution for $\Delta x$.

We illustrate the robustness of the method by plotting the MAP estimate of the effectiveness of the treatment in Fig 5(a). As before, the estimate only shows small random perturbations, as expected in any experiment with a stochastic nature and is to be contrasted with the plots in Fig 5(b) which show the changes in the two blinding indexes throughout the experiment. Again, with the change in the participants' auxiliary data, both indexes also change in value. It is insightful to observe that unlike in Exp 3, in this instance the values of the two indexes do not exhibit agreement on the direction of change of the level of blinding. This reflects the importance that the auxiliary data interpretation plays in the methods of both James *et al.* and Bang *et al.*

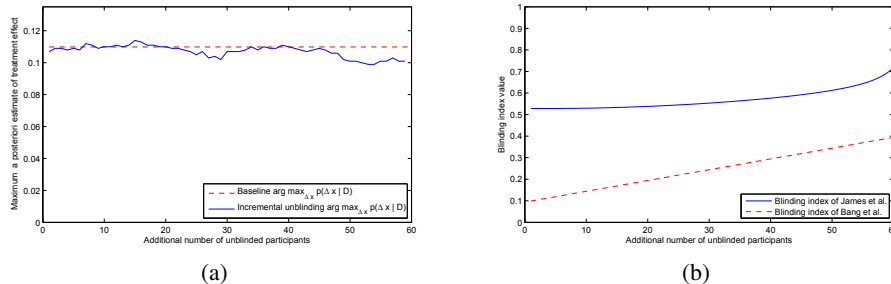

| | |
|:---:|:---:|
| (a) | (b) |

Figure 5: Exp 4: (a) The MAP estimate of the treatment effectiveness as the participants assigned to both the treatment and the control groups are progressively unblinded. (b) The values of the blinding indexes $\rho_1$ (blue line) and $\rho_2'$ (red line), computed at each step of the progressive unblinding.

## 6   Discussion

**Degenerate Cases**   One of the key ideas behind the present method is that it is meaningful to compare only the sub-groups matched by their auxiliary responses. While a greater number of subgroups may provide more precise auxilliary/blinding information, the introduced partitioning of data decreases the statistical strength of each comparison of the corresponding sub-groups. In an extreme case, a particular sub-group may be empty. In other words, it is possible that none of the participants of the treatment or the control group expressed a particular belief regarding their treatment assignment. Although this may appear as a problem at first, a more careful examination of such cases reveals that this is not so.

Firstly, note that whenever at least one pair of matching sub-groups is non-empty, the proposed method is able to compute a meaningful estimate of differential treatment effectiveness. In instances when there are no non-empty matching sub-groups, the nature of degeneracy can provide useful insight to the clinician. The absence of individuals in $G_{T+}$ may indicate that the participants assigned to the treatment group have either been poorly blinded but misidentified the received treatment, or that the treatment was vastly ineffective and was recognized as such by the participants assigned to it. Similarly, the absence of individuals in $G_{T-}$ may indicate that the participants assigned to the treatment group have either been poorly blinded and correctly identified the received treatment, or that the treatment was obviously effective. In all cases, because degenerate data is trivial to recognize, the clinician is immediately made aware of the presence of a major flaw in the experimental design. The cause of degeneration can then be determined using the knowledge of the administered interventions, and the statistics of both auxiliary responses and trial outcomes.

**Further Insight**   In Sec 4.2 we derived posteriors corresponding both to only a single pair of corresponding sub-groups in Eq (7) and to the entirety of data, that is, all sub-groups in Eq (9). While the latter of these is of primary interest, the clinician can derive further useful insight into the nature of studied treatment by comparative examination of sub-group posteriors too.

The least interesting case is when the sub-group posteriors and the total posterior exhibit similar characteristics (e.g. the location of the mode). However, consider the case when that is not so. For example, let us say that the posterior corresponding to the two matching "don't know" subgroups has the mode near $\Delta x \approx 0$ and the total posterior has a decidedly positive mode (with suitably small standard deviations, to make the observation statistically significant). This could indicate that there may be so-called "non-responders" in the treatment group, i.e. individuals which did not respond positively to the treatment which in most people does produce a positive result [4, 8]. Similar arguments can be made by considering differences between other sub-group posteriors. Ultimately, the exact interpretation is in the hands of the clinicians who should use their insight into the nature of the administered interventions to infer further information of this type.

## 7   Summary and Conclusions

This paper examined the problem of assessing the extent of blindness in a clinical trial. We demonstrated a series of fundamental flaws in blinding index based approaches and thus proposed a novel framework. At the centre of our idea is that the comparison of the treatment and control groups should be done in like-for-like fashion, giving rise to the partitioning of participants into sub-groups, each sub-group sharing the same intervention and post-trial responses. A Bayesian framework was used to interpret jointly the auxiliary and trial outcome data, giving the clinician a meaningful and readily understandable end result. The effectiveness of our method was demonstrated empirically in a simulation study, which showed its robustness in a variety of scenarios.

# References

[1] H. Bang, L. Ni, and C. E. Davis. Assessment of blinding in clinical trials. *Contemp Clin Trials*, 25(2):143–156, 2004.

[2] H. K. Beecher. The powerful placebo. *JAMA*, 159(17):1602–1606, 1955.

[3] F. Benedetti, H. S. Mayberg, T. D. Wager, C. S. Stohler, and J.-K. Zubieta. Neurobiological mechanisms of the placebo effect. *J Neurosci*, 25(45):10390–10402, 2005.

[4] G. Costantino, F. Furfaro, A. Belvedere, A. Alibrandi, and W. Fries. Thiopurine treatment in inflammatory bowel disease: Response predictors, safety, and withdrawal in follow-up. *J Crohns Colitis*, 2011.

[5] H. Hemilä. Assessment of blinding may be inappropriate after the trial. *Contemp Clin Trials*, 26(4):512–514, 2005.

[6] H.-H. Henneicke-von Zepelin. Letter to the editor. *Contemp Clin Trials*, 26(4):512, 2005.

[7] K. E. James, D. A. Bloch, K. K. Lee, H. C. Kraemer, and R. K. Fuller. An index for assessing blindness in a multi-centre clinical trial: disulfiram for alcohol cessation–a va cooperative study. *Stat Med*, 15(13):1421–1434, 1996.

[8] D. Karakitsos, J. Papanikolaou, A. Karabinis, R. Alalawi, M. Wachtel, C. Jumper, D. Alexopoulos, and P. Davlouros. Acute effect of sildenafil on central hemodynamics in mechanically ventilated patients with WHO group III pulmonary hypertension and right ventricular failure necessitating administration of dobutamine. *Int J Cardiol*, 2012.

[9] H. S. Mayberg, J. A. Silva, S. K. Brannan, J. L. Tekell, R. K. Mahurin, S. McGinnis, and P. A. Jerabek. The functional neuroanatomy of the placebo effect. 159:728–737, 2002.

[10] D. E. Moerman and W. B. Jonas. Deconstructing the placebo effect and finding the meaning response. *Ann Intern Med*, 136(6):471–476, 2002.

[11] G. H. Montgomery and I. Kirsch. Classical conditioning and the placebo effect. *Pain*, 72(1–2):107–113, 1997.

